# An Analog VLSI Chip for Finding Edges from Zero-crossings

**Wyeth Bair    Christof Koch**
Computation and Neural Systems Program
Caltech 216-76
Pasadena, CA 91125

## Abstract

We have designed and tested a one-dimensional 64 pixel, analog CMOS VLSI chip which localizes intensity edges in real-time. This device exploits on-chip photoreceptors and the natural filtering properties of resistive networks to implement a scheme similar to and motivated by the Difference of Gaussians (DOG) operator proposed by Marr and Hildreth (1980). Our chip computes the zero-crossings associated with the difference of two exponential weighting functions. If the derivative across this zero-crossing is above a threshold, an edge is reported. Simulations indicate that this technique will extend well to two dimensions.

## 1    INTRODUCTION

The zero-crossings of the Laplacian of the Gaussian, $\nabla^2 G$, are often used for detecting edges. Marr and Hildreth (1980) argued that the Mexican-hat shape of the $\nabla^2 G$ operator can be approximated by the difference of two Gaussians (DOG). In this spirit, we have built a chip that takes the difference of two resistive-network smoothings of photoreceptor input and finds the resulting zero-crossings. The Green's function of the resistive network, a symmetrical decaying exponential, differs from the Gaussian filter. Figure 1 shows the "Mexican-hat" shape of the DOG superimposed on the "witch-hat" shape of the difference of exponentials (DOE) filter implemented by our chip.

This implementation has the particular advantage of exploiting the smoothing operation performed by a linear resistive network, shown in Figure 2. In such a network, data voltages $d$ are applied to the nodes along the network via conductances $G$, and the nodes are connected by resistances $R$. Following Kirchhoff's laws, the network

node voltages $v$ settle to values such that power dissipation is minimized. One may think of the network node voltages $v$ as the convolution of the input with the symmetrical decaying exponential filter function. The characteristic length of this filter function is approximately $1/\sqrt{RG}$, where $G$ is the data conductance and $R$ the network resistance.

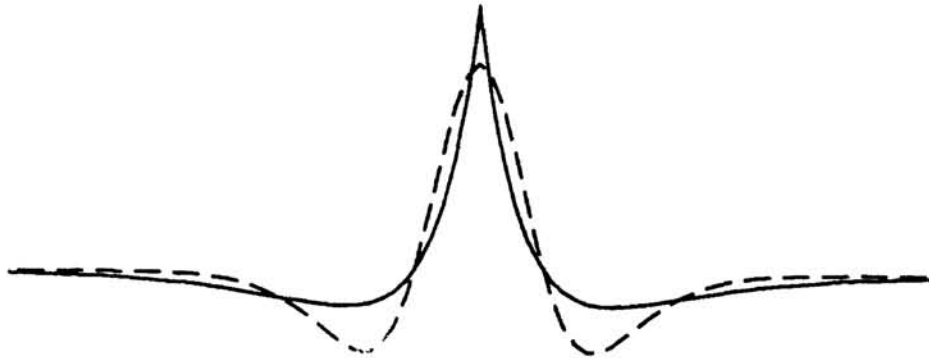

Figure 1: The Mexican-hat shape of the difference of Gaussians (dotted) and the witch-hat shape of the filter implemented by our chip.

Such a network is easily implemented in silicon and avoids the burden of additional circuitry which others have used to implement Gaussian kernels. Our simulations with digitized camera images show only minor differences between the zero-crossings from the DOE filter and those from the DOG.

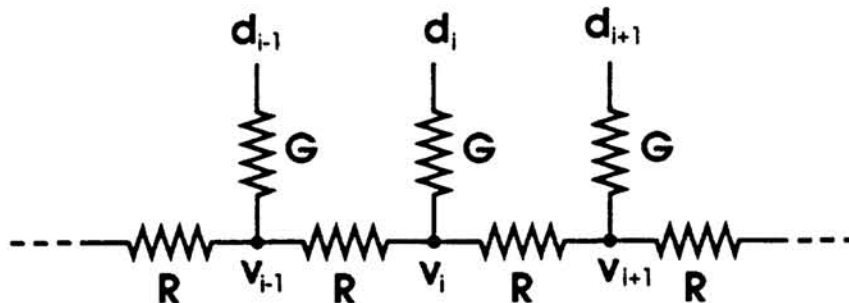

Figure 2: 1-D resistive network.

## 2   ANALOG VLSI IMPLEMENTATION

This chip was implemented with a $2.0\,\mu$m CMOS n-well process available through the MOSIS silicon foundry. Intensity edges are detected using four stages of circuitry: photoreceptors capture incoming light, a pair of 1-D resistive networks smooth the input image, transconductance amplifiers subtract the smoothed images, and digital circuitry detects zero-crossings. Figures 3 and 4 show block diagrams for two pixels of the 64 pixel chip.

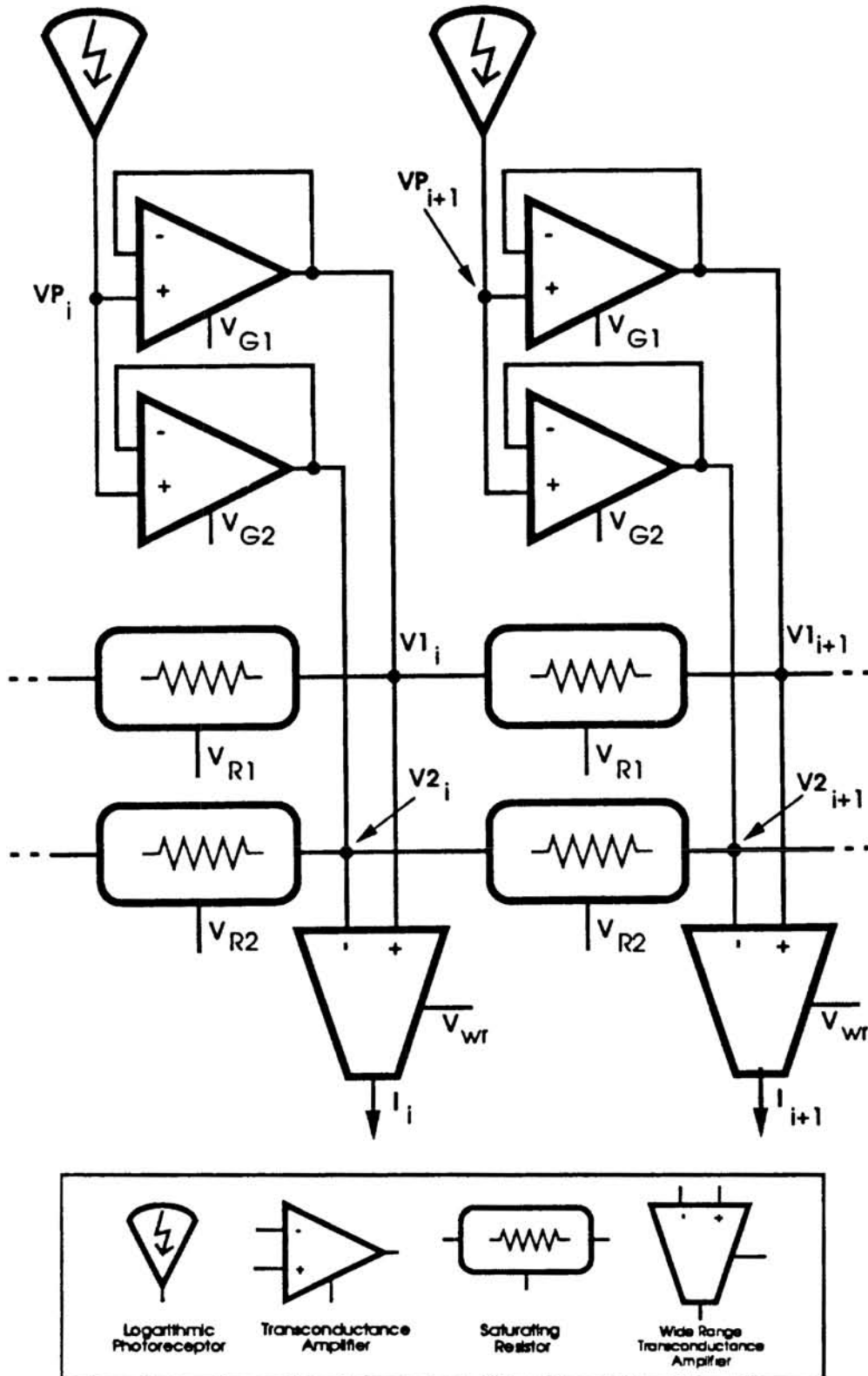

Figure 3: Block circuit diagram for two of 64 pixels as described in Section 2.

Processing begins at a line of photoreceptors spaced $100\mu m$ apart which encode the logarithm of light intensity as a voltage $VP$, shown in Figure 3. The set of voltages from the photoreceptors are reported to corresponding nodes of two resistive networks via transconductance amplifiers connected as followers. The followers' voltage biases, $V_{G1}$ and $V_{G2}$, can be adjusted off-chip to independently set the data conductances for each resistive network. The network resistors are implemented as Mead's saturating resistors (Mead, 1989). Voltage biases $V_{R1}$ and $V_{R2}$ allow independent off-chip adjustment of the two network resistances. The data conductance and network resistance values determine the space constant of the smoothing filter which each network implements. The sets of voltages $V1$ and $V2$, shown in Figure 3, represent the two filtered versions of the image. Wide-range transconductance amplifiers (Mead, 1989) produce currents, $I$, proportional to the difference $V1 - V2$.

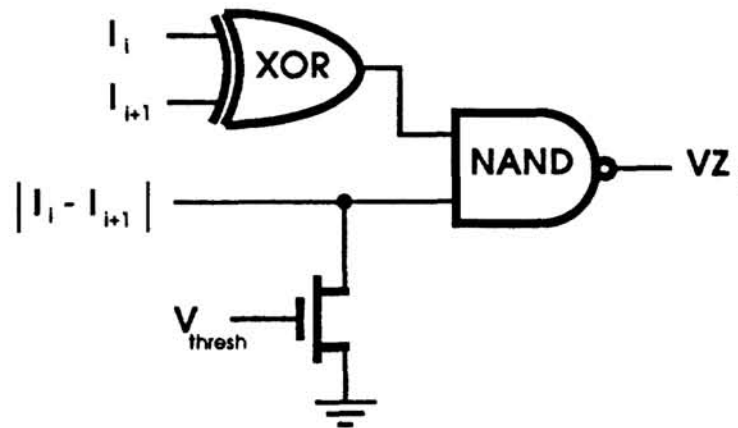

Figure 4: Zero-crossing detection and threshold circuitry.

Figure 4 shows the final stage of processing which detects zero-crossings in the sequence of currents $I$ and implements a threshold on the slope of those zero-crossings. Currents $I_i$ and $I_{i+1}$ charge or discharge the inputs of an exclusive OR gate. The output of this gate is the first input to a NAND gate which is used to implement the threshold. A current proportional to the magnitude of the difference $I_i - I_{i+1}$ charges the second input of the NAND gate, while a threshold current discharges this input. If the charging current, representing the slope of the zero-crossing, is greater than the threshold current set off-chip by the bias voltage $V_{thresh}$, this NAND input is charged to logical 1, otherwise, this input is discharged to logical 0. The output of the NAND gate, $VZ_i$ indicates the presence, logical 0, or the absence, logical 1, of a zero-crossing with slope greater than $I_{thresh}$.

A final stage of circuitry is used to multiplex the sequence of 63 bits, $VZ$, and corresponding currents $I_i - I_{i+1}$ indicating the slope of the zero-crossings.

# 3  BEHAVIOR

We tested the behavior of the chip by placing a small lens above the silicon wafer to focus an image onto the array of photoreceptors. The input light profile that we used is shown in Figure 5a. Figure 5b is an oscilloscope trace showing the smoothed voltages ($V1$ and $V2$ of Figure 3) corresponding to the filtered versions of the image. The difference of these two smoothed voltage traces is shown in Figure 5c. Arrows indicate the locations of two zero-crossings which the chip reports at the output. The reported zero-crossings accurately localize the positions of the edges in the image. The trace in Figure 5c crosses zero at other locations, but zero-crossings with slope less than the adjustable threshold are masked by the circuitry shown in Figure 4. This allows for noise and imperfections in the circuitry and can be used to filter out weaker edges which are not relevant to the application.

Figure 6 shows the response when two fingers are held one meter from the lens and swept across the field of view. The fingers appear as bright regions against a darker background. The chip accurately localizes the four edges (two per finger) as indicated by the pulses below each voltage trace. As the fingers move quickly back and forth across the field of view, the image and the zero-crossings follow the object with no perceived delay. The measured response time of the chip to the appearance of a detectable discontinuity in light intensity varies from about $100\,\mu$sec in bright indoor illumination to about 10msec in a dark room. The time constant is longer for lower illumination due to the design of the logarithmic photoreceptor (Mead, 1989).

The chip has been proven to be a reliable and robust edge detector through its use in two systems. It provides data for a system designed at the Hughes Aircraft Artificial Intelligence Center which tracks edges and reports their velocities at over 300Hz. Also, we have built a hand-held battery powered device which displays the locations of edges on a bank of 63 LEDs. This device accurately detects edges in many different environments, ranging from a dimly lit room to bright outdoor sunlight.

# 4  SIMULATIONS OF A 2-D VERSION

We have used a computer simulation of rectangular networks of ideal linear resistors to test the extension of this technique in two dimensions. Results indicate that the zero-crossings from the difference of two symmetrical exponential filters are qualitatively similar to those from the DOG. Figure 7 compares the zero-crossing from a difference of Gaussians filter (left) to those from a difference of resistive networks filter (right). For the DOG, a Gaussian of $\sigma = 1.25$ pixels is subtracted from a Gaussian of $\sigma = 0.75$ pixels. For the resistive networks, a filter of characteristic length 1 was subtracted from one with characteristic length $1/\sqrt{2}$. Weaker zero-crossings are masked from both output images by thresholding on the slope to emphasize comparison of the stronger edges.

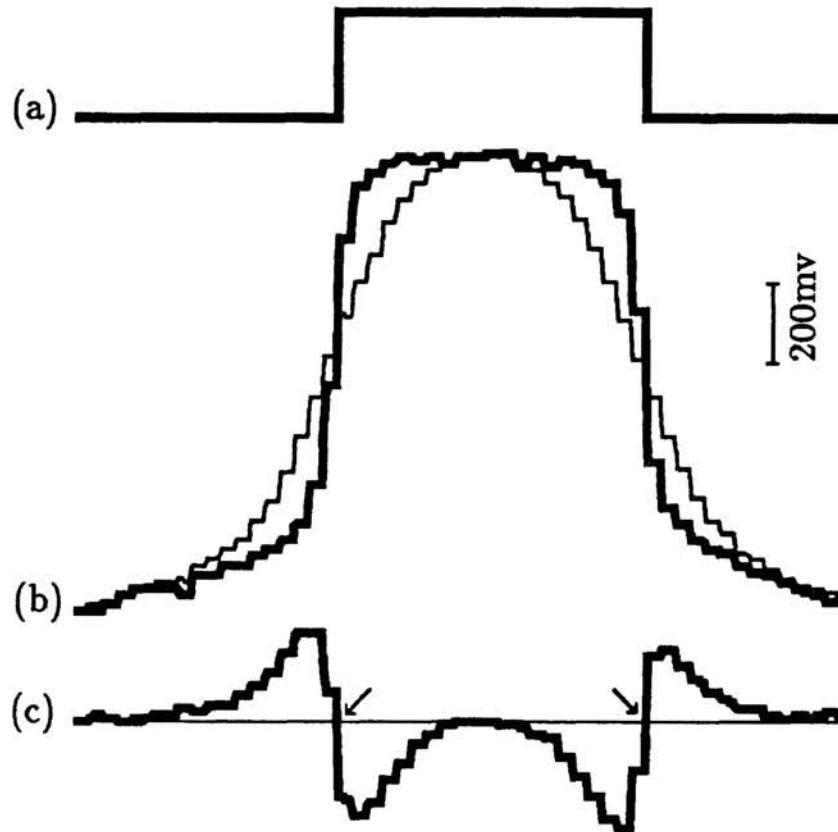

Figure 5: Chip response to a light bar stimulus.

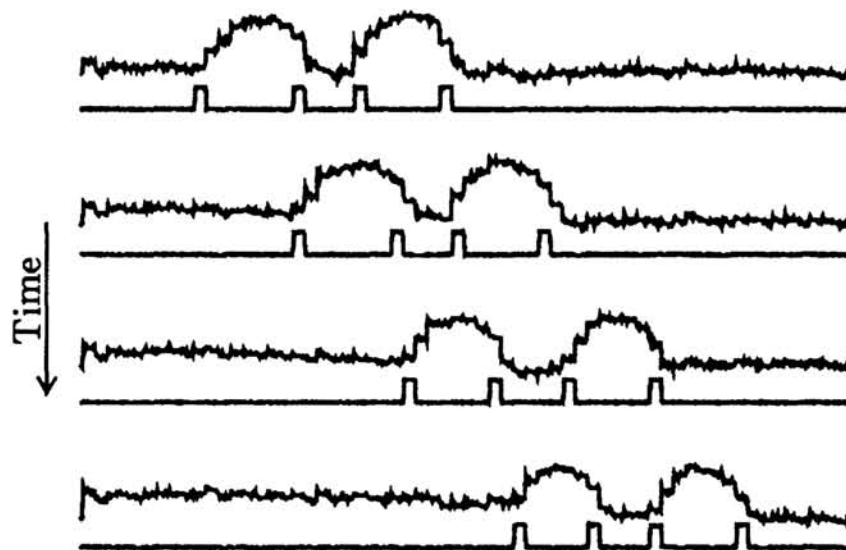

Figure 6: Chip response to two moving stimuli.

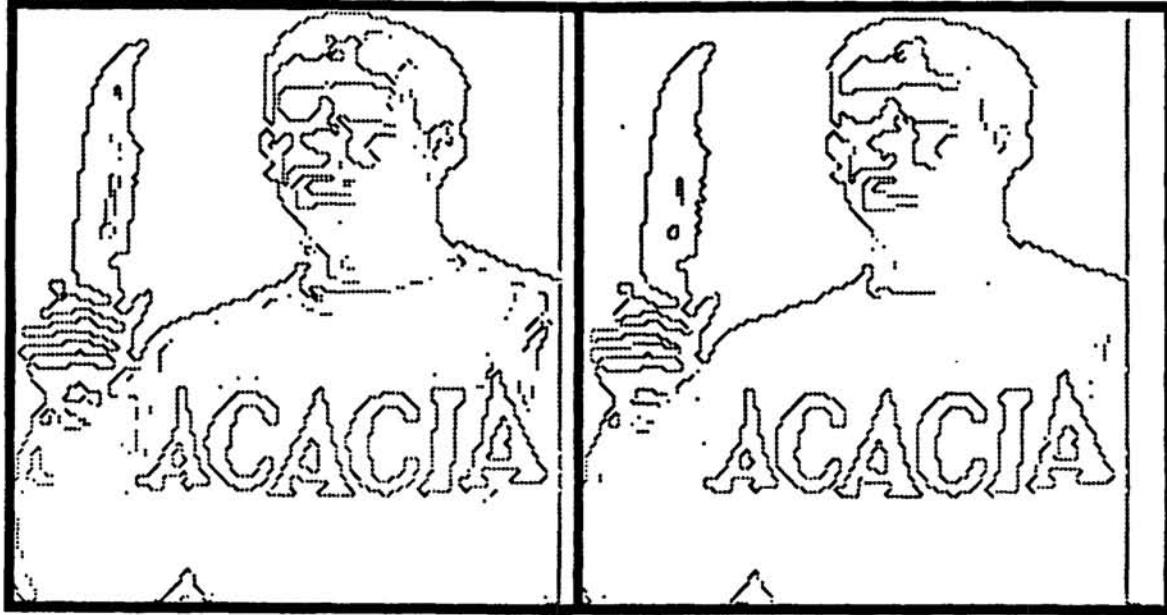

Figure 7: Zero-crossings from the difference of two Gaussians (left) and similar output from a difference of decaying exponentials (right).

## 5   CONCLUSION

Our analog VLSI chip shows that finding the thresholded zero-crossings of the difference of exponential filters is a robust technique for localizing intensity edges in real-time. The robust behavior of the chip in systems to track edges and determine velocity demonstrates the usefulness of implementing simple algorithms in analog VLSI and the advantages of avoiding large, more general digital systems for these purposes.

**Acknowledgements**

Many thanks to Carver Mead. Our laboratory is partially supported by grants from the Office of Naval Research, the Rockwell International Science Center and the Hughes Aircraft Artificial Intelligence Center. Wyeth Bair is supported by a National Science Foundation Graduate Fellowship. Thanks also to Steve DeWeerth and John Harris.

**References**

Marr, D. and Hildreth, E.C. (1980) Theory of edge detection. *Proc. Roy. Soc. Lond.* B 207:187-217.

Mead, C.A. (1989) *Analog VLSI and Neural Systems.* Addison-Wesley: Reading, MA.